# Distance Metric Learning for Large Margin Nearest Neighbor Classification

**Kilian Q. Weinberger, John Blitzer and Lawrence K. Saul**
Department of Computer and Information Science, University of Pennsylvania
Levine Hall, 3330 Walnut Street, Philadelphia, PA 19104
{kilianw, blitzer, lsaul}@cis.upenn.edu

## Abstract

We show how to learn a Mahanalobis distance metric for $k$-nearest neighbor (kNN) classification by semidefinite programming. The metric is trained with the goal that the $k$-nearest neighbors always belong to the same class while examples from different classes are separated by a large margin. On seven data sets of varying size and difficulty, we find that metrics trained in this way lead to significant improvements in kNN classification—for example, achieving a test error rate of 1.3% on the MNIST handwritten digits. As in support vector machines (SVMs), the learning problem reduces to a convex optimization based on the hinge loss. Unlike learning in SVMs, however, our framework requires no modification or extension for problems in multiway (as opposed to binary) classification.

## 1   Introduction

The $k$-nearest neighbors (kNN) rule [3] is one of the oldest and simplest methods for pattern classification. Nevertheless, it often yields competitive results, and in certain domains, when cleverly combined with prior knowledge, it has significantly advanced the state-of-the-art [1, 14]. The kNN rule classifies each unlabeled example by the majority label among its $k$-nearest neighbors in the training set. Its performance thus depends crucially on the *distance metric* used to identify nearest neighbors.

In the absence of prior knowledge, most kNN classifiers use simple Euclidean distances to measure the dissimilarities between examples represented as vector inputs. Euclidean distance metrics, however, do not capitalize on any statistical regularities in the data that might be estimated from a large training set of labeled examples.

Ideally, the distance metric for kNN classification should be adapted to the particular problem being solved. It can hardly be optimal, for example, to use the same distance metric for face recognition as for gender identification, even if in both tasks, distances are computed between the same fixed-size images. In fact, as shown by many researchers [2, 6, 7, 8, 12, 13], kNN classification can be significantly improved by learning a distance metric from labeled examples. Even a simple (global) linear transformation of input features has been shown to yield much better kNN classifiers [7, 12]. Our work builds in a novel direction on the success of these previous approaches.

In this paper, we show how to learn a Mahanalobis distance metric for kNN classification. The metric is optimized with the goal that *k-nearest neighbors always belong to the same class while examples from different classes are separated by a large margin.* Our goal for metric learning differs in a crucial way from those of previous approaches that minimize the pairwise distances between *all* similarly labeled examples [12, 13, 17]. This latter objective is far more difficult to achieve and does not leverage the full power of kNN classification, whose accuracy does *not* require that all similarly labeled inputs be tightly clustered.

Our approach is largely inspired by recent work on neighborhood component analysis [7] and metric learning by energy-based models [2]. Though based on the same goals, however, our methods are quite different. In particular, we are able to cast our optimization as an instance of semidefinite programming. Thus the optimization we propose is convex, and its global minimum can be efficiently computed.

Our approach has several parallels to learning in support vector machines (SVMs)—most notably, the goal of margin maximization and a convex objective function based on the hinge loss. In light of these parallels, we describe our approach as *large margin nearest neighbor* (LMNN) classification. Our framework can be viewed as the logical counterpart to SVMs in which kNN classification replaces linear classification.

Our framework contrasts with classification by SVMs, however, in one intriguing respect: it requires no modification for problems in multiway (as opposed to binary) classification. Extensions of SVMs to multiclass problems typically involve combining the results of many binary classifiers, or they require additional machinery that is elegant but non-trivial [4]. In both cases the training time scales at least linearly in the number of classes. By contrast, our learning problem has no explicit dependence on the number of classes.

## 2  Model

Let $\{(\vec{x}_i, y_i)\}_{i=1}^n$ denote a training set of $n$ labeled examples with inputs $\vec{x}_i \in \mathcal{R}^d$ and discrete (but not necessarily binary) class labels $y_i$. We use the binary matrix $y_{ij} \in \{0, 1\}$ to indicate whether or not the labels $y_i$ and $y_j$ match. Our goal is to learn a linear transformation $\mathbf{L} : \mathcal{R}^d \to \mathcal{R}^d$, which we will use to compute squared distances as:

$$\mathcal{D}(\vec{x}_i, \vec{x}_j) = \|\mathbf{L}(\vec{x}_i - \vec{x}_j)\|^2. \tag{1}$$

Specifically, we want to learn the linear transformation that optimizes kNN classification when distances are measured in this way. We begin by developing some useful terminology.

**Target neighbors**
In addition to the class label $y_i$, for each input $\vec{x}_i$ we also specify $k$ "target" neighbors—that is, $k$ other inputs with the same label $y_i$ that we wish to have minimal distance to $\vec{x}_i$, as computed by eq. (1). In the absence of prior knowledge, the target neighbors can simply be identified as the $k$ nearest neighbors, determined by Euclidean distance, that share the same label $y_i$. (This was done for all the experiments in this paper.) We use $\eta_{ij} \in \{0, 1\}$ to indicate whether input $\vec{x}_j$ is a target neighbor of input $\vec{x}_i$. Like the binary matrix $y_{ij}$, the matrix $\eta_{ij}$ is fixed and does not change during learning.

**Cost function**
Our cost function over the distance metrics parameterized by eq. (1) has two competing terms. The first term penalizes large distances between each input and its target neighbors, while the second term penalizes small distances between each input and all other inputs that do not share the same label. Specifically, the cost function is given by:

$$\varepsilon(\mathbf{L}) = \sum_{ij} \eta_{ij} \|\mathbf{L}(\vec{x}_i - \vec{x}_j)\|^2 + c \sum_{ijl} \eta_{ij} (1 - y_{il}) \left[ 1 + \|\mathbf{L}(\vec{x}_i - \vec{x}_j)\|^2 - \|\mathbf{L}(\vec{x}_i - \vec{x}_l)\|^2 \right]_+,$$
$$\tag{2}$$

where in the second term $[z]_+ = \max(z, 0)$ denotes the standard hinge loss and $c > 0$ is some positive constant (typically set by cross validation). Note that the first term only penalizes large distances between inputs and target neighbors, *not between all similarly labeled examples*.

**Large margin**

The second term in the cost function incorporates the idea of a margin. In particular, for each input $\vec{x}_i$, the hinge loss is incurred by differently labeled inputs whose distances do not exceed, by one absolute unit of distance, the distance from input $\vec{x}_i$ to any of its target neighbors. The cost function thereby favors distance metrics in which differently labeled inputs maintain a large margin of distance and do not threaten to "invade" each other's neighborhoods. The learning dynamics induced by this cost function are illustrated in Fig. 1 for an input with $k=3$ target neighbors.

**Parallels with SVMs**

The competing terms in eq. (2) are analogous to those in the cost function for SVMs [11]. In both cost functions, one term penalizes the norm of the "parameter" vector (i.e., the weight vector of the maximum margin hyperplane, or the linear transformation in the distance metric), while the other incurs the hinge loss for examples that violate the condition of unit margin. Finally, just as the hinge loss in SVMs is only triggered by examples near the decision boundary, the hinge loss in eq. (2) is only triggered by differently labeled examples that invade each other's neighborhoods.

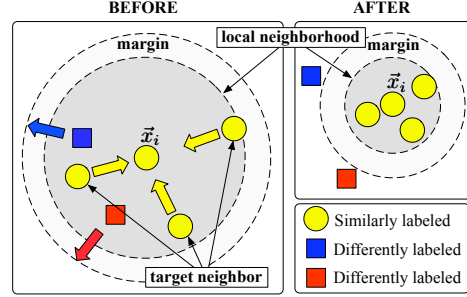

Figure 1: Schematic illustration of one input's neighborhood $\vec{x}_i$ before training (*left*) versus after training (*right*). The distance metric is optimized so that: (i) its $k=3$ target neighbors lie within a smaller radius after training; (ii) differently labeled inputs lie outside this smaller radius, with a margin of at least one unit distance. Arrows indicate the gradients on distances arising from the optimization of the cost function.

**Convex optimization**

We can reformulate the optimization of eq. (2) as an instance of semidefinite programming [16]. A semidefinite program (SDP) is a linear program with the additional constraint that a matrix whose elements are linear in the unknown variables is required to be positive semidefinite. SDPs are convex; thus, with this reformulation, the global minimum of eq. (2) can be efficiently computed. To obtain the equivalent SDP, we rewrite eq. (1) as:

$$\mathcal{D}(\vec{x}_i, \vec{x}_j) = (\vec{x}_i - \vec{x}_j)^\top \mathbf{M} (\vec{x}_i - \vec{x}_j), \tag{3}$$

where the matrix $\mathbf{M} = \mathbf{L}^\top \mathbf{L}$, parameterizes the Mahalanobis distance metric induced by the linear transformation $\mathbf{L}$. Rewriting eq. (2) as an SDP in terms of $\mathbf{M}$ is straightforward, since the first term is already linear in $\mathbf{M} = \mathbf{L}^\top \mathbf{L}$ and the hinge loss can be "mimicked" by introducing slack variables $\xi_{ij}$ for all pairs of differently labeled inputs (i.e., for all $\langle i, j \rangle$ such that $y_{ij} = 0$). The resulting SDP is given by:

---

**Minimize** $\sum_{ij} \eta_{ij} (\vec{x}_i - \vec{x}_j)^\top \mathbf{M} (\vec{x}_i - \vec{x}_j) + c \sum_{ij} \eta_{ij} (1 - y_{il}) \xi_{ijl}$ **subject to:**

**(1)** $(\vec{x}_i - \vec{x}_l)^\top \mathbf{M} (\vec{x}_i - \vec{x}_l) - (\vec{x}_i - \vec{x}_j)^\top \mathbf{M} (\vec{x}_i - \vec{x}_j) \geq 1 - \xi_{ijl}$

**(2)** $\xi_{ijl} \geq 0$

**(3)** $\mathbf{M} \succeq 0.$

---

The last constraint $\mathbf{M} \succeq 0$ indicates that the matrix $\mathbf{M}$ is required to be positive semidefinite. While this SDP can be solved by standard online packages, general-purpose solvers

tend to scale poorly in the number of constraints. Thus, for our work, we implemented our own special-purpose solver, exploiting the fact that most of the slack variables $\{\xi_{ij}\}$ never attain positive values[1]. The slack variables $\{\xi_{ij}\}$ are sparse because most labeled inputs are well separated; thus, their resulting pairwise distances do not incur the hinge loss, and we obtain very few *active* constraints. Our solver was based on a combination of sub-gradient descent in both the matrices $\mathbf{L}$ and $\mathbf{M}$, the latter used mainly to verify that we had reached the global minimum. We projected updates in $\mathbf{M}$ back onto the positive semidefinite cone after each step. Alternating projection algorithms provably converge [16], and in this case our implementation worked much faster than generic solvers[2].

## 3 Results

We evaluated the algorithm in the previous section on seven data sets of varying size and difficulty. Table 1 compares the different data sets. Principal components analysis (PCA) was used to reduce the dimensionality of image, speech, and text data, both to speed up training and avoid overfitting. Except for Isolet and MNIST, all of the experimental results are averaged over several runs of randomly generated 70/30 splits of the data. Isolet and MNIST have pre-defined training/test splits. For the other data sets, we randomly generated 70/30 splits for each run. Both the number of target neighbors ($k$) and the weighting parameter ($c$) in eq. (2) were set by cross validation. (For the purpose of cross-validation, the training sets were further partitioned into training and validation sets.) We begin by reporting overall trends, then discussing the individual data sets in more detail.

We first compare kNN classification error rates using Mahalanobis versus Euclidean distances. To break ties among different classes, we repeatedly reduced the neighborhood size, ultimately classifying (if necessary) by just the $k=1$ nearest neighbor. Fig. 2 summarizes the main results. Except on the smallest data set (where over-training appears to be an issue), the Mahalanobis distance metrics learned by semidefinite programming led to significant improvements in kNN classification, both in training and testing. The training error rates reported in Fig. 2 are leave-one-out estimates.

We also computed test error rates using a variant of kNN classification, inspired by previous work on energy-based models [2]. Energy-based classification of a test example $\vec{x}_t$ was done by finding the label that minimizes the cost function in eq. (2). In particular, for a hypothetical label $y_t$, we accumulated the squared distances to the $k$ nearest neighbors of $\vec{x}_t$ that share the same label in the training set (corresponding to the first term in the cost function); we also accumulated the hinge loss over all pairs of differently labeled examples that result from labeling $\vec{x}_t$ by $y_t$ (corresponding to the second term in the cost function). Finally, the test example was classified by the hypothetical label that minimized the combination of these two terms:

$$y_t = \text{argmin}_{y_t} \sum_j \eta_{tj} \|\mathbf{L}(\vec{x}_t - \vec{x}_j)\|^2 + c \sum_{j,i=t \vee l=t} \eta_{ij}(1-y_{il})\left[1 + \|\mathbf{L}(\vec{x}_i - \vec{x}_j)\|^2 - \|\mathbf{L}(\vec{x}_i - \vec{x}_l)\|^2\right]_+$$

As shown in Fig. 2, energy-based classification with this assignment rule generally led to even further reductions in test error rates.

Finally, we compared our results to those of multiclass SVMs [4]. On each data set (except MNIST), we trained multiclass SVMs using linear and RBF kernels; Fig. 2 reports the results of the better classifier. On MNIST, we used a non-homogeneous polynomial kernel of degree four, which gave us our best results. (See also [9].)

|  | Iris | Wine | Faces | Bal | Isolet | News | MNIST |
|---|---|---|---|---|---|---|---|
| examples (train) | 106 | 126 | 280 | 445 | 6238 | 16000 | 60000 |
| examples (test) | 44 | 52 | 120 | 90 | 1559 | 2828 | 10000 |
| classes | 3 | 3 | 40 | 3 | 26 | 20 | 10 |
| input dimensions | 4 | 13 | 1178 | 4 | 617 | 30000 | 784 |
| features after PCA | 4 | 13 | 30 | 4 | 172 | 200 | 164 |
| constraints | 5278 | 7266 | 78828 | 76440 | 37 Mil | 164 Mil | 3.3 Bil |
| active constraints | 113 | 1396 | 7665 | 3099 | 45747 | 732359 | 243596 |
| CPU time (per run) | 2s | 8s | 7s | 13s | 11m | 1.5h | 4h |
| runs | 100 | 100 | 100 | 100 | 1 | 10 | 1 |

Table 1: Properties of data sets and experimental parameters for LMNN classification.

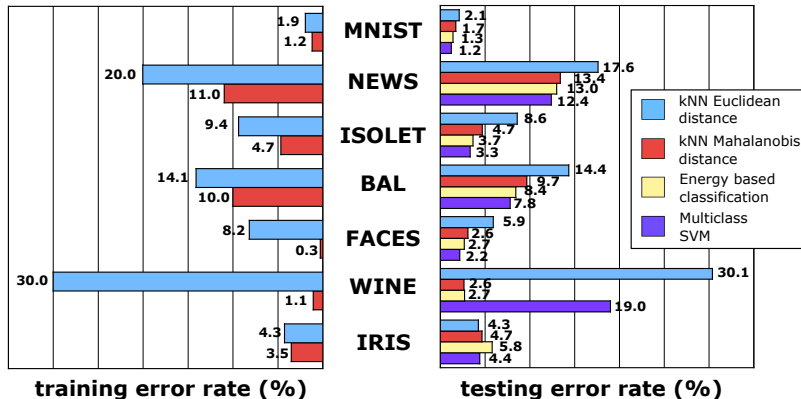

Figure 2: Training and test error rates for kNN classification using Euclidean versus Mahalanobis distances. The latter yields lower test error rates on all but the smallest data set (presumably due to over-training). Energy-based classification (see text) generally leads to further improvement. The results approach those of state-of-the-art multiclass SVMs.

**Small data sets with few classes**
The wine, iris, and balance data sets are small data sets, with less than 500 training examples and just three classes, taken from the UCI Machine Learning Repository[3]. On data sets of this size, a distance metric can be learned in a matter of seconds. The results in Fig. 2 were averaged over 100 experiments with different random 70/30 splits of each data set. Our results on these data sets are roughly comparable (i.e., better in some cases, worse in others) to those of neighborhood component analysis (NCA) and relevant component analysis (RCA), as reported in previous work [7].

**Face recognition**
The AT&T face recognition data set[4] contains 400 grayscale images of 40 individuals in 10 different poses. We downsampled the images from to $38 \times 31$ pixels and used PCA to obtain 30-dimensional eigenfaces [15]. Training and test sets were created by randomly sampling 7 images of each person for training and 3 images for testing. The task involved 40-way classification—essentially, recognizing a face from an unseen pose. Fig. 2 shows the improvements due to LMNN classification. Fig. 3 illustrates the improvements more graphically by showing how the $k = 3$ nearest neighbors change as a result of learning a Mahalanobis metric. (Though the algorithm operated on low dimensional eigenfaces, for clarity the figure shows the rescaled images.)

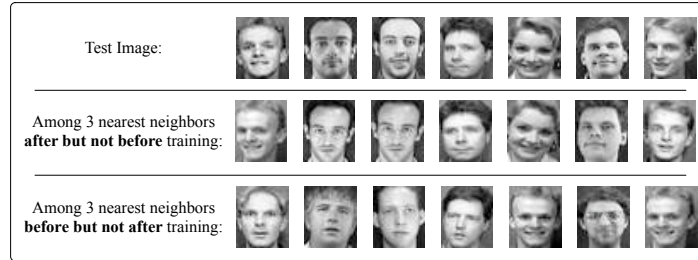

Figure 3: Images from the AT&T face recognition data base. *Top row*: an image correctly recognized by kNN classification ($k = 3$) with Mahalanobis distances, but not with Euclidean distances. *Middle row*: correct match among the $k=3$ nearest neighbors according to Mahalanobis distance, but not Euclidean distance. *Bottom row*: incorrect match among the $k=3$ nearest neighbors according to Euclidean distance, but not Mahalanobis distance.

**Spoken letter recognition**

The Isolet data set from UCI Machine Learning Repository has 6238 examples and 26 classes corresponding to letters of the alphabet. We reduced the input dimensionality (originally at 617) by projecting the data onto its leading 172 principal components—enough to account for 95% of its total variance. On this data set, Dietterich and Bakiri report test error rates of $4.2\%$ using nonlinear backpropagation networks with 26 output units (one per class) and $3.3\%$ using nonlinear backpropagation networks with a 30-bit error correcting code [5]. LMNN with energy-based classification obtains a test error rate of $3.7\%$.

**Text categorization**

The 20-newsgroups data set consists of posted articles from 20 newsgroups, with roughly 1000 articles per newsgroup. We used the 18828-version of the data set[5] which has cross-postings removed and some headers stripped out. We tokenized the newsgroups using the rainbow package [10]. Each article was initially represented by the weighted word-counts of the 20,000 most common words. We then reduced the dimensionality by projecting the data onto its leading 200 principal components. The results in Fig. 2 were obtained by averaging over 10 runs with 70/30 splits for training and test data. Our best result for LMNN on this data set at $13.0\%$ test error rate improved significantly on kNN classification using Euclidean distances. LMNN also performed comparably to our best multiclass SVM [4], which obtained a $12.4\%$ test error rate using a linear kernel and 20000 dimensional inputs.

**Handwritten digit recognition**

The MNIST data set of handwritten digits[6] has been extensively benchmarked [9]. We deskewed the original $28 \times 28$ grayscale images, then reduced their dimensionality by retaining only the first 164 principal components (enough to capture 95% of the data's overall variance). Energy-based LMNN classification yielded a test error rate at 1.3%, cutting the baseline kNN error rate by over one-third. Other comparable benchmarks [9] (not exploiting additional prior knowledge) include multilayer neural nets at 1.6% and SVMs at 1.2%. Fig. 4 shows some digits whose nearest neighbor changed as a result of learning, from a mismatch using Euclidean distance to a match using Mahanalobis distance.

## 4  Related Work

Many researchers have attempted to learn distance metrics from labeled examples. We briefly review some recent methods, pointing out similarities and differences with our work.

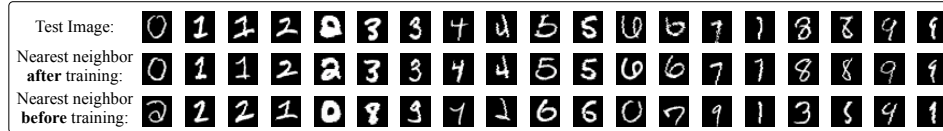

Figure 4: *Top row:* Examples of MNIST images whose nearest neighbor changes during training. *Middle row:* nearest neighbor after training, using the Mahalanobis distance metric. *Bottom row:* nearest neighbor before training, using the Euclidean distance metric.

Xing et al [17] used semidefinite programming to learn a Mahalanobis distance metric for clustering. Their algorithm aims to minimize the sum of squared distances between similarly labeled inputs, while maintaining a lower bound on the sum of distances between differently labeled inputs. Our work has a similar basis in semidefinite programming, but differs in its focus on local neighborhoods for kNN classification.

Shalev-Shwartz et al [12] proposed an online learning algorithm for learning a Mahalanobis distance metric. The metric is trained with the goal that all similarly labeled inputs have small pairwise distances (bounded from above), while all differently labeled inputs have large pairwise distances (bounded from below). A margin is defined by the difference of these thresholds and induced by a hinge loss function. Our work has a similar basis in its appeal to margins and hinge loss functions, but again differs in its focus on local neighborhoods for kNN classification. In particular, we do not seek to minimize the distance between all similarly labeled inputs, only those that are specified as neighbors.

Goldberger et al [7] proposed neighborhood component analysis (NCA), a distance metric learning algorithm especially designed to improve kNN classification. The algorithm minimizes the probability of error under stochastic neighborhood assignments using gradient descent. Our work shares essentially the same goals as NCA, but differs in its construction of a convex objective function.

Chopra et al [2] recently proposed a framework for similarity metric learning in which the metrics are parameterized by pairs of identical convolutional neural nets. Their cost function penalizes large distances between similarly labeled inputs and small distances between differently labeled inputs, with penalties that incorporate the idea of a margin. Our work is based on a similar cost function, but our metric is parameterized by a linear transformation instead of a convolutional neural net. In this way, we obtain an instance of semidefinite programming.

Relevant component analysis (RCA) constructs a Mahalanobis distance metric from a weighted sum of in-class covariance matrices [13]. It is similar to PCA and linear discriminant analysis (but different from our approach) in its reliance on second-order statistics.

Hastie and Tibshirani [**?**] and Domeniconi et al [6] consider schemes for locally adaptive distance metrics that vary throughout the input space. The latter work appeals to the goal of margin maximization but otherwise differs substantially from our approach. In particular, Domeniconi et al [6] suggest to use the decision boundaries of SVMs to induce a locally adaptive distance metric for kNN classification. By contrast, our approach (though similarly named) does not involve the training of SVMs.

## 5 Discussion

In this paper, we have shown how to learn Mahalanobis distance metrics for kNN classification by semidefinite programming. Our framework makes no assumptions about the structure or distribution of the data and scales naturally to large number of classes. Ongoing

work is focused in three directions. First, we are working to apply LMNN classification to problems with hundreds or thousands of classes, where its advantages are most apparent. Second, we are investigating the kernel trick to perform LMNN classification in nonlinear feature spaces. As LMMN already yields highly nonlinear decision boundaries in the original input space, however, it is not obvious that "kernelizing" the algorithm will lead to significant further improvement. Finally, we are extending our framework to learn locally adaptive distance metrics [6, 8] that vary across the input space. Such metrics should lead to even more flexible and powerful large margin classifiers.

## Footnotes

[1] A great speedup can be achieved by solving an SDP that only monitors a fraction of the margin conditions, then using the resulting solution as a starting point for the actual SDP of interest.

[2] A matlab implementation is currently available at http://www.seas.upenn.edu/~kilianw/lmnn.

[3]Available at http://www.ics.uci.edu/∼mlearn/MLRepository.html.

[4]Available at http://www.uk.research.att.com/facedatabase.html

[5] Available at http://people.csail.mit.edu/jrennie/20Newsgroups/

[6] Available at http://yann.lecun.com/exdb/mnist/

# References

[1] S. Belongie, J. Malik, and J. Puzicha. Shape matching and object recognition using shape contexts. *IEEE Transactions on Pattern Analysis and Machine Intelligence (PAMI)*, 24(4):509–522, 2002.

[2] S. Chopra, R. Hadsell, and Y. LeCun. Learning a similiarty metric discriminatively, with application to face verification. In *Proceedings of the IEEE Conference on Computer Vision and Pattern Recognition (CVPR-05)*, San Diego, CA, 2005.

[3] T. Cover and P. Hart. Nearest neighbor pattern classification. In *IEEE Transactions in Information Theory, IT-13*, pages 21–27, 1967.

[4] K. Crammer and Y. Singer. On the algorithmic implementation of multiclass kernel-based vector machines. *Journal of Machine Learning Research*, 2:265–292, 2001.

[5] T. G. Dietterich and G. Bakiri. Solving multiclass learning problems via error-correcting output codes. In *Journal of Artificial Intelligence Research*, number 2 in 263-286, 1995.

[6] C. Domeniconi, D. Gunopulos, and J. Peng. Large margin nearest neighbor classifiers. *IEEE Transactions on Neural Networks*, 16(4):899–909, 2005.

[7] J. Goldberger, S. Roweis, G. Hinton, and R. Salakhutdinov. Neighbourhood components analysis. In L. K. Saul, Y. Weiss, and L. Bottou, editors, *Advances in Neural Information Processing Systems 17*, pages 513–520, Cambridge, MA, 2005. MIT Press.

[8] T. Hastie and R. Tibshirani. Discriminant adaptive nearest neighbor classification. *IEEE Transactions on Pattern Analysis and Machine Intelligence (PAMI)*, 18:607–616, 1996.

[9] Y. LeCun, L. Jackel, L. Bottou, A. Brunot, C. Cortes, J. Denker, H. Drucker, I. Guyon, U. Muller, E. Sackinger, P. Simard, and V. Vapnik. A comparison of learning algorithms for handwritten digit recognition. In F.Fogelman and P.Gallinari, editors, *Proceedings of the 1995 International Conference on Artificial Neural Networks (ICANN-95)*, pages 53–60, Paris, 1995.

[10] A. K. McCallum. Bow: A toolkit for statistical language modeling, text retrieval, classification and clustering. http://www.cs.cmu.edu/ mccallum/bow, 1996.

[11] B. Schölkopf and A. J. Smola. *Learning with Kernels: Support Vector Machines, Regularization, Optimization, and Beyond*. MIT Press, Cambridge, MA, 2002.

[12] S. Shalev-Shwartz, Y. Singer, and A. Y. Ng. Online and batch learning of pseudo-metrics. In *Proceedings of the 21st International Conference on Machine Learning*, Banff, Canada, 2004.

[13] N. Shental, T. Hertz, D. Weinshall, and M. Pavel. Adjustment learning and relevant component analysis. In *Proceedings of the Seventh European Conference on Computer Vision (ECCV-02)*, volume 4, pages 776–792, London, UK, 2002. Springer-Verlag.

[14] P. Y. Simard, Y. LeCun, and J. Decker. Efficient pattern recognition using a new transformation distance. In *Advances in Neural Information Processing Systems*, volume 6, pages 50–58, San Mateo, CA, 1993. Morgan Kaufman.

[15] M. Turk and A. Pentland. Eigenfaces for recognition. *Journal of Cognitive Neuroscience*, 3(1):71–86, 1991.

[16] L. Vandenberghe and S. P. Boyd. Semidefinite programming. *SIAM Review*, 38(1):49–95, March 1996.

[17] E. P. Xing, A. Y. Ng, M. I. Jordan, and S. Russell. Distance metric learning, with application to clustering with side-information. In T. G. Dietterich, S. Becker, and Z. Ghahramani, editors, *Advances in Neural Information Processing Systems 14*, Cambridge, MA, 2002. MIT Press.
